# Self-organisation in real neurons: Anti-Hebb in 'Channel Space'?

**Anthony J. Bell**
AI-lab,
Vrije Universiteit Brussel
Pleinlaan 2, B-1050 Brussels
BELGIUM, (tony@arti.vub.ac.be)

## Abstract

Ion channels are the dynamical systems of the nervous system. Their distribution within the membrane governs not only communication of information between neurons, but also how that information is integrated within the cell. Here, an argument is presented for an 'anti-Hebbian' rule for changing the distribution of *voltage-dependent* ion channels in order to flatten voltage curvatures in dendrites. Simulations show that this rule can account for the self-organisation of dynamical receptive field properties such as resonance and direction selectivity. It also creates the conditions for the faithful conduction within the cell of signals to which the cell has been exposed. Various possible cellular implementations of such a learning rule are proposed, including activity-dependent migration of channel proteins in the plane of the membrane.

## 1 INTRODUCTION

### 1.1 NEURAL DYNAMICS

Neural inputs and outputs are temporal, but there are no established ways to think about temporal learning and dynamical receptive fields. The currently popular simple recurrent nets have only one kind of dynamical component: a *capacitor*, or time constant. Though it is *possible* to create any kind of dynamics using capacitors and static non-linearities, it is also *possible* to write any program on a Turing machine.

Biological evolution, it seems, has elected for diversity and complexity over uniformity and simplicity in choosing voltage-dependent ion channels as the 'instruction set' for dynamical computation.

## 1.2  ION CHANNELS

As more ion channels with varying kinetics are discovered, the question of their computational role has become more pertinent. Figure 1, derived from a model thalamic cell, shows the log time constants of 11 currents, plotted against the voltage ranges over which they activate or inactivate. The variety of available kinetics is probably under-represented here since a combinatorial number of differences can be obtained by combining different protein sub-domains to make a channel [6].

Given the likelihood that channels are inhomogenously distributed throughout the dendrites [7], one way to tackle the question of their computational role is to search for a self-organisational principle for forming this distribution. Such a 'learning rule' could be construed as operating during development or dynamically during the life of an organism, and could be considered complementary to learning involving synaptic changes. The resulting distribution and mix of channels would then be, in some sense, optimal for integrating and communicating the particular high-dimensional spatio-temporal inputs which the cell was accustomed to receiving.

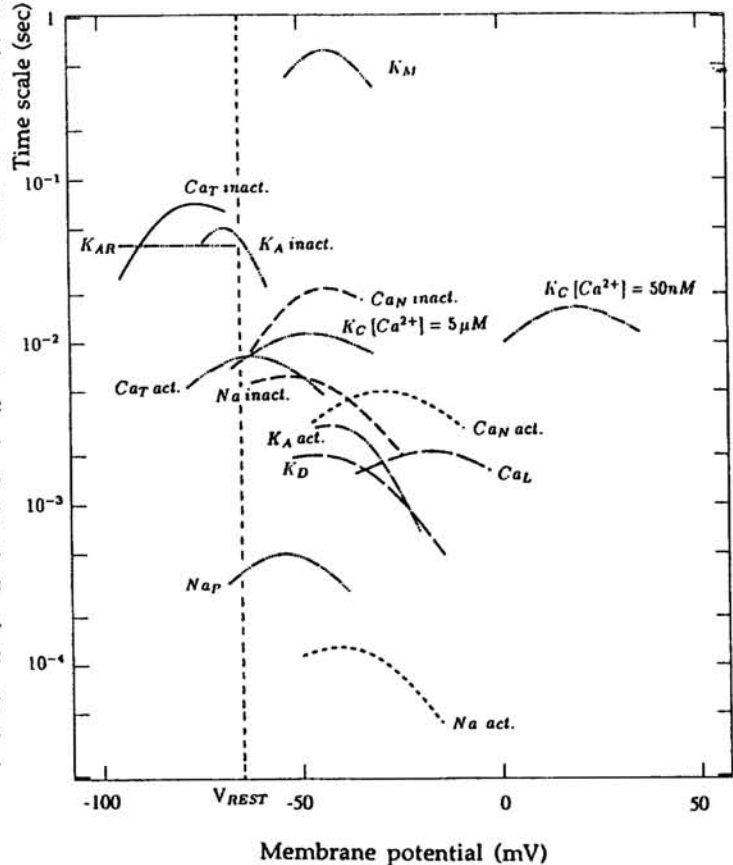

Figure 1: Diversity of ion channel kinetics. The voltage-dependent equilibrium log time constants of 11 channels are plotted here for the voltage ranges for which their activation (or inactivation) variables go from $0.1 \rightarrow 0.9$ (or $0.9 \rightarrow 0.1$). The channel kinetics are taken from a model by W.Lytton [10]. Notice the range of speeds of operation from the spiking $Na^+$ channel around 0.1ms, to the $K_M$ channel in the 1s (cognitive) range.

## 2  THE BIOPHYSICAL SUBSTRATE

The substrate for self-organisation is the standard cable model for a dendrite or axon:

$$G_a \frac{\partial^2 V}{\partial x^2} = C \frac{\partial V}{\partial t} + \sum_j \overline{G}_j g_j (V - E_j) + \sum_k \overline{G}_k g_k (V - E_k) \qquad (1)$$

In this $G_a$ represents the conductance along the axis of the cable, $C$ is the capacitance and the two sums represent *synaptic* (indexed by $j$) and *intrinsic* (indexed by $k$) currents. $\overline{G}$ is a maximum conductance (a channel density or 'weight'), $g$ is the time-varying fraction of the conductance active, and $E$ is a reversal potential. The system can be summarised by saying that the current flow out of a segment of a neuron is equal to the sum of currents input to that segment, plus the capacitive charging of the membrane.

This leads to a simpler form:

$$i = \sum_j \overline{g}_j i_j + \sum_k \overline{g}_k i_k \qquad (2)$$

Here, $i = \partial^2 V/\partial x^2$, $\overline{g}_j = \overline{G}_j/G_a$, $i_j = g_j(V - E_j)$ and $C$ is considered as an intrinsic conductance whose $\overline{g}_k$ and $i_k$ are $C/G_a$ and $\partial V/\partial t$ respectively. In this form, it is more clear that each part of a neuron can be considered as a 'unit', diffusively coupled to its neighbours, to which it passes its weighted sum of inputs. The weights

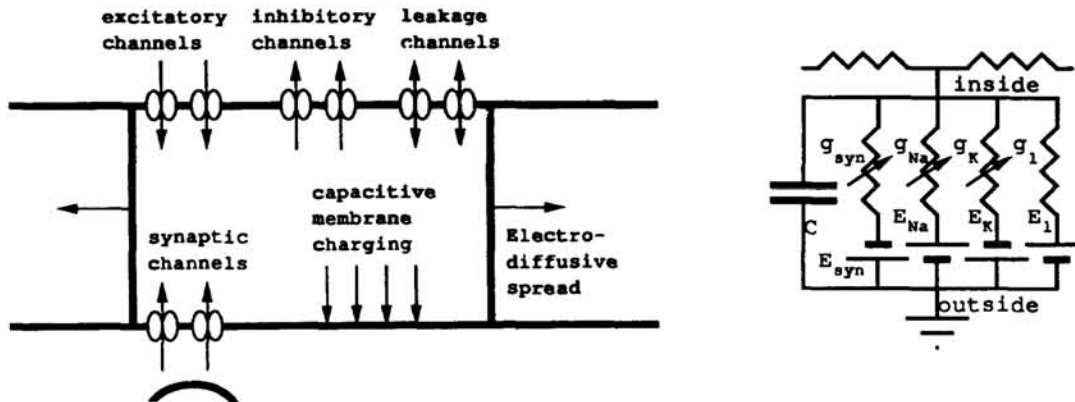

Figure 1: A compartment of a neuron, shown schematically and as a circuit. The cable equation is just Kirchoff's Law: *current in = current out*

$\overline{g}_k$, representing the $G_a$-normalised densities of channel species $k$, are considered to span *channel space*, as opposed to the $\overline{g}_j$ weights which are our standard synaptic strength parameters. Parameters determining the dynamics of $g_k$'s specify points in *kinetics space*. Neuromodulation [8], a universally important phenomenon in real nervous systems, consists of specific chemicals inducing short-term changes in the kinetics space co-ordinates of a channel type, resulting, for example, in shifts in the curves in Figure 1.

## 3   THE ARGUMENT FOR ANTI-HEBB

Learning algorithms, of the type successful in static systems, have not been considered for these low-level dynamical components (though see [2] for approaches to synaptic learning in realistic systems). Here, we address the issue of unsupervised learning for channel densities. In the neural network literature, unsupervised learning consists of Hebbian-type algorithms and information theoretic approaches based on objective functions [1]. In the absence of a good information theoretic framework for continuous time, non-Gaussian analog systems where noise is undefined, we resort to exploring the implications of the effects of simple local rules.

The most obvious rule following from equation 2 would be a correlational one of the following form, with the learning rate $\epsilon$ positive or negative:

$$\Delta \bar{g}_k = \epsilon i_k i \qquad (3)$$

While a polarising (or Hebbian) rule (see Figure 3) makes sense for synaptic channels as an a method for amplifying input signals, it makes less sense for intrinsic channels. Were it to operate on such channels, statistical fluctuations from the uniform channel distribution would give rise to self-reinforcing 'hot-spots' with no underlying 'signal' to amplify. For this reason, we investigate the utility of a rectifying (or anti-Hebbian) rule.

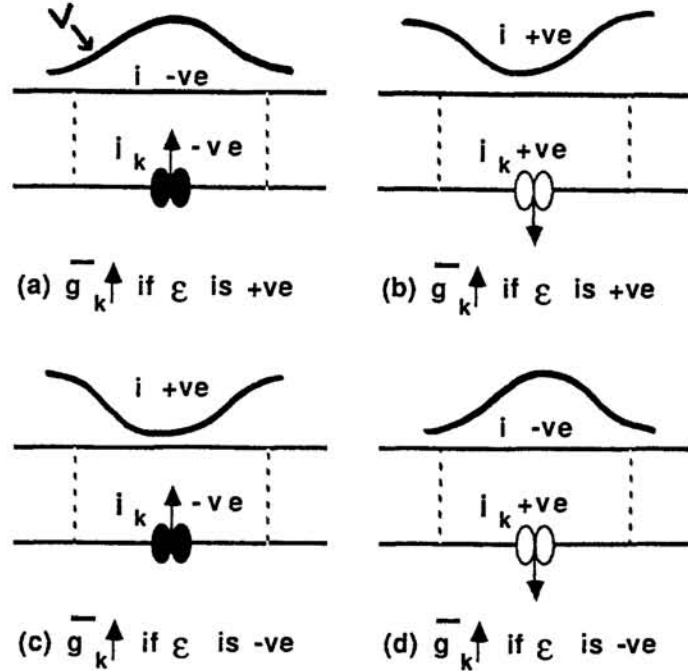

Figure 3: A schematic display showing contingent positive and negative voltage curvatures ($\pm i$) above a segment of neuron, and inward and outward currents ($\pm i_k$), through a particular channel type. In situations (a) and (b), a Hebbian version of Equation 3 will raise the channel density ($\bar{g}_k \uparrow$), and in (c) and (d) an anti-Hebbian rule will do this. In the first two cases, the channels are *polarising* the membrane potential, creating high voltage curvature, while in the latter two, they are *rectifying* (or flattening) it. Depending on the sign of $\epsilon$, equation 3 attempts to either maximise or minimise $(\partial^2 V/\partial x^2)^2$.

## 4  EXAMPLES

For the purposes of demonstration, linear RLC electrical components are often used here. These simple 'intrinsic' (non-synaptic) components have the most tractable kinetics of any, and as shown by [11] and [9], the impedances they create capture some of the properties of active membrane. The components are leakage resistances, capacitances and inductances, whose $\bar{g}_k$'s are given by $1/R$, $C$ and $1/L$ respectively. During learning, all $\bar{g}_k$'s were kept above zero for reasons of stability.

### 4.1  LEARNING RESONANCE

In this experiment, an RLC 'compartment' with no frequency preference was stimulated at a certain frequency and trained according to equation 3 with $\epsilon$ negative. After training, the frequency response curve of the circuit had a resonant peak at the training frequency (Figure 4). This result is significant since many auditory and tactile sensory cells are tuned to certain frequencies, and we know that a major component of the tuning is electrical, with resonances created by particular balances of ion channel populations [13].

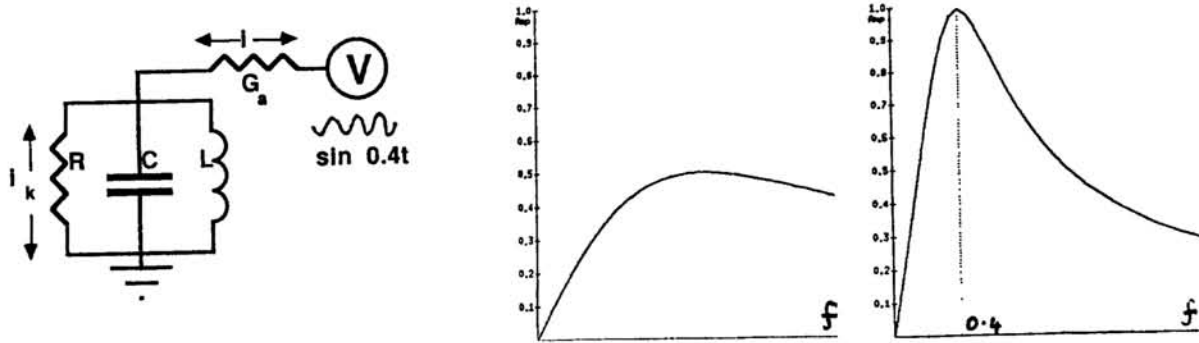

Figure 4: Learning resonance. The curves show the frequency-response curves of the compartment before and after training at a frequency of 0.4.

## 4.2   LEARNING CONDUCTION

Another role that intrinsic channels must play within a cell is the faithful transmission of information. Any voltage curvatures at a point away from a synapse signify a net cross membrane current which can be seen as distorting the signal in the cable. Thus, by removing voltage curvatures, we preserve the signal. This is demonstrated

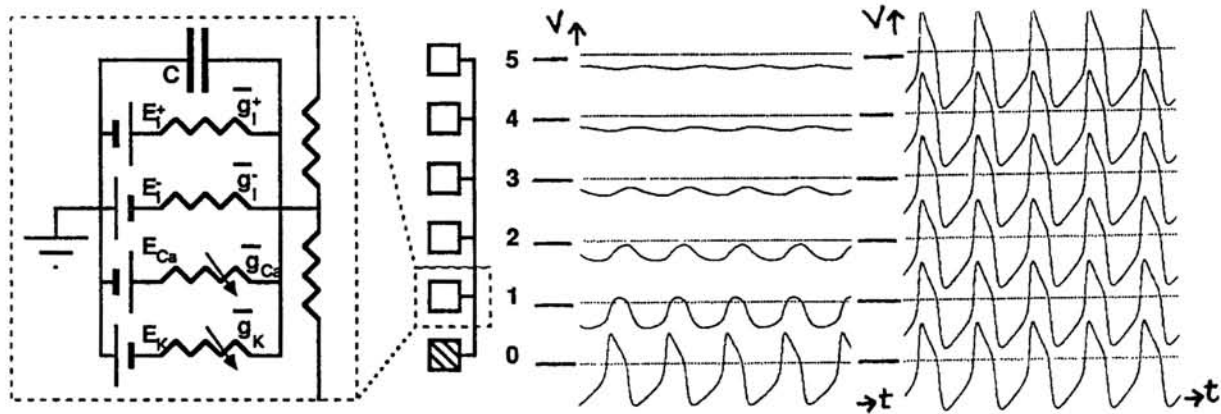

Figure 5: Learning conduction. The cable consists of a chain of compartments, which only conduct the impulse after they acquire active channels.

in the following example: 'learning to be an axon'. A non-linear spiking compartment with Morris-Lecar $Ca/K$ kinetics (see [14]) is coupled to a long passive cable. Before learning, the signal decays passively in the cable (Figure 5). The driving compartment $\bar{g}$-vector, and the capacitances in the cable are then clamped to stop the system from converging on the null solution ($\bar{g} \to 0$). All other $\bar{g}$'s (including spiking conductances in the cable) can then learn. The first thing learnt was that the inward and outward leakage conductances ($\bar{g}_l^+$ and $\bar{g}_l^-$) adjusted themselves to make the average voltage curvature in each compartment zero (just as bias units in error correction algorithms adjust to make the average error zero). Then the cable filled out with Morris-Lecar channels ($\bar{g}_{Ca}$ and $\bar{g}_K$) in *exactly the same ratios* as the driving compartment, resulting in a cable that faithfully propagated the signal.

### 4.3   LEARNING PHASE-SHIFTING (DIRECTION SELECTIVITY)

The last example involves 4 'sensory' compartments coupled to a 'somatic' compartment as in Figure 6. All are similar to the linear compartments in the resonance example except that the sensory ones receive 'synaptic' input in the form of a sinusoidal current source. The relative phases of the input were shifted to simulate left-to-right motion. After training, the 'dendritic' components had learned, using their capacitors and inductors, to cancel the phase shifts so that the inputs were synchronised in their effect on the 'soma'. This creates a large response in the trained direction, and a small one in the 'null' direction, as the phases cancelled each other.

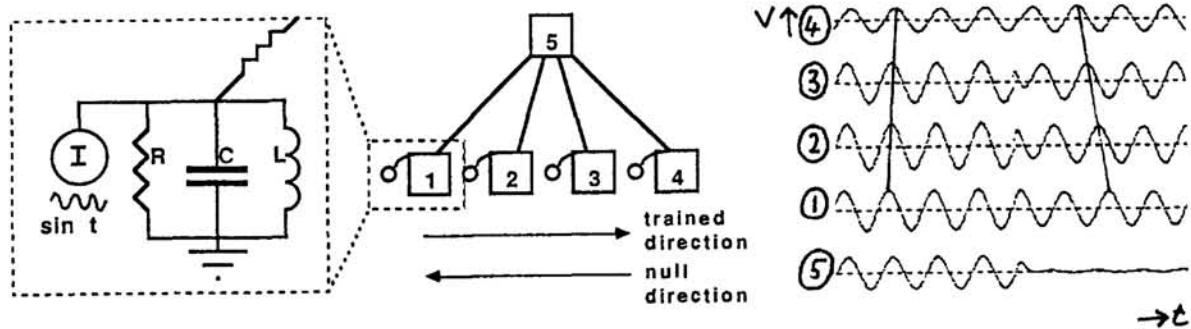

Figure 6: Learning direction selectivity. After training on a drifting sine wave, the output compartment oscillates for the trained direction but not for the null direction (see the trace, where the direction of motion is reversed halfway).

## 5   DISCUSSION

### 5.1   CELLULAR MECHANISMS

There is substantial evidence in cell biology for targeting of proteins to specific parts of the membrane, but the fact that equation 3 is dependent on the *correlation* of channel species' activity and local voltages leaves only 4 possible biological implementations:

1. the cellular targeting machinery knows what kind of channel it is delivering, and thus knows where to put it

2. channels in the wrong place are degraded faster than those in the right place

3. channels migrate to the right place while in the membrane

4. the *effective* channel density is altered by activity-dependent neuromodulation or channel-blockage

The third is perhaps the most intriguing. The diffusion of channels in the plane of the membrane, under the influence of induced electric fields has received both theoretical [4, 12] and empirical [7, 3] attention. To a first approximation, the

evolution of channel densities can be described by a Smoluchowski equation:

$$\frac{\partial \bar{g}_k}{\partial t} = a \frac{\partial^2 \bar{g}_k}{\partial x^2} + b \frac{\partial}{\partial x} \left( \bar{g}_k \frac{\partial V}{\partial x} \right) \tag{4}$$

where $a$ is the coefficient of *thermal* diffusion and $b$ is the coefficient of *field induced* motion. This system has been studied previously [4] to explain receptor-clustering in synapse formation, but if the sign of $b$ is reversed, then it fits more closely with the anti-Hebbian rule discussed here. The crucial requirement for true activity-dependence, though, is that $b$ should be different when the channel is open than when it is closed. This may be plausible since channel gating involves movements of charges across the membrane. Coefficients of thermal diffusion have been measured and found not to exceed $10^{-9}$ cm/sec. This would be enough to fine-tune channel distributions, but not to transport them all the way down dendrites.

The second method in the list is also an attractive possibility. The half-life of membrane proteins can be as low as several hours [3], and it is known that proteins can be differentially labeled for recycling [5].

## 5.2   ENERGY AND INFORMATION

The anti-Hebbian rule changes $\bar{g}_k$'s in order to minimise the square membrane current density, integrated over the cell in units of axial conductance. This corresponds in two senses to a minimisation of energy. From a circuit perspective, the energy dissipated in the axial resistances is minimised. From a metabolic perspective, the ATP used in pumping ions back across the membrane is minimised. The computation consists of minimising the expected value of this energy, given particular spatiotemporal synaptic input (assuming no change in $\bar{g}_j$'s). More precisely, it searches for:

$$Min_{(\bar{g}_k)} \left( E \left[ \int i^2(x,t) \, dG_a(x) \, dt \, \bigg| \, g_j(x,t), \, \forall j \right] \right) \tag{5}$$

This search creates mutual information between input dynamics and intrinsic dynamics. In addition, since the Laplacian ($\nabla_x^2 V = 0$) is what a diffusive system seeks to converge to anyway, the learning rule simply configures the system to speed this convergence on frequently experienced inputs.

Simple zero-energy solutions exist for the above, for example the 'ultra-leaky' compartment ($\bar{g}_l \rightarrow \infty$) and the 'point' (or non-existent) compartment ($\bar{g}_k \rightarrow 0, \, \forall k$), for compartments with and without synapses respectively. The anti-Hebb rule alone will eventually converge to such solutions, unless, for example, the leakage or capacitance are prevented from learning. Another solution (which has been successfully used for the direction selectivity example) is to make the total available quantity of each $\bar{g}_k$ finite. The $\bar{g}_k$ can then diffuse about between compartments, following the voltage gradients in a manner suggested by equation 4. The resulting behaviour is a finite-resource version of equation 3.

The next goal of this work is to produce a rigorous information theoretic account of single neuron computation. This is seen as a pre-requisite to understanding both neural coding and the computational capabilities of neural circuits, and as a step on the way to properly dynamical neural nets.

## Acknowledgements

This work was supported by a Belgian government IMPULS contract and by ES-PRIT Basic Research Action 3234. Thanks to Prof. L. Steels for his support and to Prof T. Sejnowski his hospitality at the Salk Institute where some of this work was done.

## References

[1] Becker S. 1990. Unsupervised learning procedures for neural networks, *Int. J. Neur. Sys.*

[2] Brown T., Mainen Z. et al. 1990. in *NIPS 3*, 39-45.    Mel B. 1991. in *Neural Computation*, vol 4 *to appear.*

[3] Darnell J., Lodish H. & Baltimore D. 1990. *Molecular Cell Biology*, Scientific American Books

[4] Fromherz P. 1988. Self-organization of the fluid mosaic of charged channel proteins in membranes, *Proc. Natl. Acad. Sci. USA* 85, 6353-6357

[5] Hare J. 1990. Mechanisms of membrane protein turnover, *Biochim. Biophys. Acta*, 1031, 71-90

[6] Hille B. 1992. *Ionic channels of excitable membranes, 2nd edition*, Sinauer Associates Inc., Sunderland, MA

[7] Jones O. et al. 1989. *Science* 244, 1189-1193.    Lo Y-J. & Poo M-M. 1991. *Science* 254, 1019-1022.    Stollberg J. & Fraser S. 1990. *J. Neurosci.* 10, 1, 247-255.    Angelides K. 1990. *Prog. in Clin. & Biol. Res.* 343, 199-212

[8] Kaczmarek L. & Levitan I. 1987. *Neuromodulation*, Oxford Univ. Press

[9] Koch C. 1984. Cable theory in neurons with active linearized membranes, *Biol. Cybern.* 50, 15-33

[10] Lytton W. 1991. Simulations of cortical pyramidal neurons synchronized by inhibitory interneurons *J. Neurophysiol.* 66, 3, 1059-1079

[11] Mauro A. Conti F. Dodge F. & Schor R. 1970. Subthreshold behaviour and phenomenological impedance of the giant squid axon, *J. Gen. Physiol.* 55, 497-523

[12] Poo M-M. & Young S. 1990. Diffusional and electrokinetic redistribution at the synapse: a physicochemical basis of synaptic competition, *J. Neurobiol.* 21, 1, 157-168

[13] Puil E. et al. *J. Neurophysiol.* 55, 5.    . Ashmore J.F. & Attwell D. 1985. *Proc. R. Soc. Lond. B* 226, 325-344.    Hudspeth A. & Lewis R. 1988. *J. Physiol.* 400, 275-297.

[14] Rinzel J. & Ermentrout G. 1989. Analysis of Neural Excitability and Oscillations, in Koch C. & Segev I. (eds) 1989. *Methods in Neuronal Modeling*, MIT Press